# Gaussian process regression with Student-$t$ likelihood

**Jarno Vanhatalo**
Department of Biomedical Engineering
and Computational Science
Helsinki University of Technology
Finland
jarno.vanhatalo@tkk.fi

**Pasi Jylänki**
Department of Biomedical Engineering
and Computational Science
Helsinki University of Technology
Finland
pasi.jylanki@tkk.fi

**Aki Vehtari**
Department of Biomedical Engineering
and Computational Science
Finland
Helsinki University of Technology
aki.vehtari@tkk.fi

## Abstract

In the Gaussian process regression the observation model is commonly assumed to be Gaussian, which is convenient in computational perspective. However, the drawback is that the predictive accuracy of the model can be significantly compromised if the observations are contaminated by outliers. A robust observation model, such as the Student-$t$ distribution, reduces the influence of outlying observations and improves the predictions. The problem, however, is the analytically intractable inference. In this work, we discuss the properties of a Gaussian process regression model with the Student-$t$ likelihood and utilize the Laplace approximation for approximate inference. We compare our approach to a variational approximation and a Markov chain Monte Carlo scheme, which utilize the commonly used scale mixture representation of the Student-$t$ distribution.

## 1   Introduction

A commonly used observation model in the Gaussian process (GP) regression is the Normal distribution. This is convenient since the inference is analytically tractable up to the covariance function parameters. However, a known limitation with the Gaussian observation model is its non-robustness, and replacing the normal distribution with a heavy-tailed one, such as the Student-$t$ distribution, can be useful in problems with outlying observations.

If both the prior and the likelihood are Gaussian, the posterior is Gaussian with mean between the prior mean and the observations. In conflict this compromise is not supported by either of the information sources. Thus, outlying observations may significantly reduce the accuracy of the inference. For example, a single corrupted observation may pull the posterior expectation of the unknown function value considerably far from the level described by the other observations (see Figure 1). A robust, or outlier-prone, observation model would, however, weight down the outlying observations the more, the further away they are from the other observations and prior mean.

The idea of robust regression is not new. Outlier rejection was described already by De Finetti [1] and theoretical results were given by Dawid [2], and O'Hagan [3]. Student-$t$ observation model with linear regression was studied already by West [4] and Geweke [5], and Neal [6] introduced it for GP regression. Other robust observation models include, for example, mixtures of Gaussians, Laplace

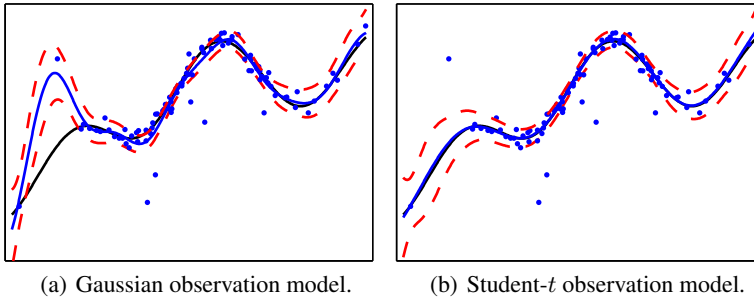

| (a) Gaussian observation model. | (b) Student-$t$ observation model. |

Figure 1: An example of regression with outliers by Neal [6]. On the left Gaussian and on the right the Student-$t$ observation model. The real function is plotted with black line.

distribution and input dependent observation models [7–10]. The challenge with the Student-$t$ model is the inference, which is analytically intractable. A common approach has been to use the scale-mixture representation of the Student-t distribution [5], which enables Gibbs sampling [5, 6], and a factorized variational approximation (VB) for the posterior inference [7, 11].

Here, we discuss the properties of the GP regression with a Student-$t$ likelihood and utilize the Laplace approximation for the approximate inference. We discuss the known weaknesses of the approximation scheme and show that in practice it works very well and quickly. We use several different data sets to compare it to both a full MCMC and a factorial VB, which utilize the scale mixture equivalent of the Student-t distribution. We show that the predictive performances are similar and that the Laplace's method approximates the posterior covariance somewhat better than VB. We also point out some of the similarities between these two methods and discuss their differences.

## 2 Robust regression with Gaussian processes

Consider a regression problem, where the data comprise observations $y_i = f(\mathbf{x}_i) + \epsilon_i$ at input locations $\mathbf{X} = \{\mathbf{x}_i\}_{i=1}^n$, where the observation errors $\epsilon_1, ..., \epsilon_n$ are zero-mean exchangeable random variables. The object of inference is the latent function $f$, which is given a Gaussian process prior. This implies that any finite subset of latent variables, $\mathbf{f} = \{f(\mathbf{x}_i)\}_{i=1}^n$, has a multivariate Gaussian distribution. In particular, at the observed input locations $\mathbf{X}$ the latent variables have a distribution

$$p(\mathbf{f}|\mathbf{X}) = \mathcal{N}(\mathbf{f}|\mu, \mathbf{K}_{\mathrm{f,f}}), \tag{1}$$

where $\mathbf{K}_{\mathrm{f,f}}$ is the covariance matrix and $\mu$ the mean function. For the notational simplicity, we will use a zero-mean Gaussian process. Each element in the covariance matrix is a realization of covariance function, $[\mathbf{K}_{\mathrm{f,f}}]_{ij} = k(\mathbf{x}_i, \mathbf{x}_j)$, which represents the prior assumptions of the smoothness of the latent function (for a detailed introduction on GP regression see [12]). The covariance function used in this work is the stationary squared exponential $k_{\mathrm{se}}(\mathbf{x}_i, \mathbf{x}_j) = \sigma_{\mathrm{se}}^2 \exp(-\sum_{d=1}^{D}(x_{i,d} - x_{j,d})^2/l_d^2)$, where $\sigma_{\mathrm{se}}^2$ is the scaling parameter and $l_d$ are the length-scales.

A formal definition of robustness is given, for example, in terms of an outlier-prone observation model. The observation model is defined to be outlier-prone of order $n$, if $p(f|y_1, ..., y_{n+1}) \rightarrow p(f|y_1, ..., y_n)$ as $y_{n+1} \rightarrow \infty$ [3, 4]. That is, the effect of a single conflicting observation to the posterior becomes asymptotically negligible as the observation approaches infinity. This contrasts heavily with the Gaussian observation model where each observation influences the posterior no matter how far it is from the others. The zero-mean Student-$t$ distribution

$$p(y_i|f_i, \sigma, \nu) = \frac{\Gamma((\nu+1)/2)}{\Gamma(\nu/2)\sqrt{\nu\pi}\sigma}\left(1 + \frac{(y_i - f_i)^2}{\nu\sigma^2}\right)^{-(\nu+1)/2}, \tag{2}$$

where $\nu$ is the degrees of freedom and $\sigma$ the scale parameter [13], is outlier prone of order 1, and it can reject up to $m$ outliers if there are at least $2m$ observations in all [3]. From this on we will collect all the hyperparameters into $\theta = \{\sigma_{\mathrm{se}}^2, l_1, ..., l_D, \sigma, \nu\}$.

# 3   Inference with the Laplace approximation

## 3.1   The conditional posterior of the latent variables

Our approach is motivated by the Laplace approximation in GP classification [14]. A similar approximation has been considered by West [4] in the case of robust linear regression and by Rue et al. [15] in their integrated nested Laplace approximation (INLA). Below we follow the notation of Rasmussen and Williams [12].

A second order Taylor expansion of $\log p(\mathbf{f} \,|\, \mathbf{y}, \theta)$ around the mode, gives a Gaussian approximation

$$p(\mathbf{f} \,|\, \mathbf{y}, \theta) \approx q(\mathbf{f} \,|\, \mathbf{y}, \theta) = N(\mathbf{f} \,|\, \hat{\mathbf{f}}, \mathbf{\Sigma}),$$

where $\hat{\mathbf{f}} = \arg\max_{\mathbf{f}} p(\mathbf{f} \,|\, \mathbf{y}, \theta)$ and $\mathbf{\Sigma}^{-1}$ is the Hessian of the negative log conditional posterior at the mode $\hat{\mathbf{f}}$ [12, 13]:

$$\mathbf{\Sigma}^{-1} = -\nabla\nabla \log p(\mathbf{f} \,|\, \mathbf{y}, \theta)|_{\mathbf{f}=\hat{\mathbf{f}}} = \mathbf{K}_{\mathrm{f,f}}^{-1} + \mathbf{W}, \tag{3}$$

where

$$\mathbf{W}_{ii} = -(\nu + 1)\frac{r_i^2 - \nu\sigma^2}{(r_i^2 + \nu\sigma^2)^2}, \tag{4}$$

$r_i = (y_i - f_i)$, and $\mathbf{W}_{ji} = 0$ if $i \neq j$.

## 3.2   The maximum a posterior estimate of the hyperparameters

To find a maximum a posterior estimate (MAP) for the hyperparameters, we write $p(\theta|\mathbf{y}) \propto p(\mathbf{y}\,|\theta)p(\theta)$, where

$$p(\mathbf{y}\,|\theta) = \int p(\mathbf{y}|\mathbf{f})p(\mathbf{f}\,|\mathbf{X},\theta)d\mathbf{f}, \tag{5}$$

is the marginal likelihood. To find an approximation, $q(\mathbf{y}\,|\theta)$, for the marginal likelihood one can utilize the Laplace method second time [12]. A Taylor expansion of the logarithm of the integrand in (5) around $\hat{\mathbf{f}}$ gives a Gaussian integral over $\mathbf{f}$ multiplied by a constant, giving

$$\log q(\mathbf{y}\,|\theta) = \log p(\mathbf{y}|\hat{\mathbf{f}}) - \frac{1}{2}\hat{\mathbf{f}}^{\mathrm{T}}\mathbf{K}_{\mathrm{f,f}}^{-1}\hat{\mathbf{f}} - \frac{1}{2}\log|\mathbf{K}_{\mathrm{f,f}}| - \frac{1}{2}\log|\mathbf{K}_{\mathrm{f,f}}^{-1} + \mathbf{W}|. \tag{6}$$

The hyperparameters can then be optimized by maximizing the approximate log marginal posterior, $\log q(\theta|\mathbf{y}) \propto \log q(\mathbf{y}\,|\theta) + \log p(\theta)$. This is differentiable with respect to $\theta$, which enables the use of gradient based optimization to find $\hat{\theta} = \arg\max_\theta q(\theta|\mathbf{y})$ [12].

## 3.3   Making predictions

The approximate posterior distribution of a latent variable $f_*$ at a new input location $\mathbf{x}_*$ is also Gaussian, and therefore defined by its mean and variance [12]

$$\mathrm{E}_{q(\mathbf{f}\,|\,\mathbf{y},\theta)}[f_*|\mathbf{X},\mathbf{y},\mathbf{x}_*] = \mathrm{K}_{*,\mathrm{f}}\,\mathbf{K}_{\mathrm{f,f}}^{-1}\,\hat{\mathbf{f}} = \mathrm{K}_{*,\mathrm{f}}\,\nabla\log p(\mathbf{y}\,|\hat{\mathbf{f}}) \tag{7}$$

$$\mathrm{Var}_{q(\mathbf{f}\,|\,\mathbf{y},\theta)}[f_*|\mathbf{X},\mathbf{y},\mathbf{x}_*] = \mathrm{K}_{*,*} - \mathrm{K}_{*,\mathrm{f}}(\mathbf{K}_{\mathrm{f,f}} + \mathbf{W}^{-1})^{-1}\,\mathrm{K}_{\mathrm{f},*}. \tag{8}$$

The predictive distribution of a new observation is obtained by marginalizing over the posterior distribution of $f_*$

$$q(y_*|\mathbf{X},\mathbf{y},\mathbf{x}_*) = \int p(y_*|f_*)q(f_*|\mathbf{X},\mathbf{y},\mathbf{x}_*)df_*, \tag{9}$$

which can be evaluated, for example, with a Gaussian quadrature integration.

## 3.4   Properties of the Laplace approximation

The Student-$t$ distribution is not log-concave, and therefore the posterior distribution may be multimodal. The immediate concern from this is that a unimodal Laplace approximation may give a poor estimate for the posterior. This is, however, a problem for all unimodal approximations,

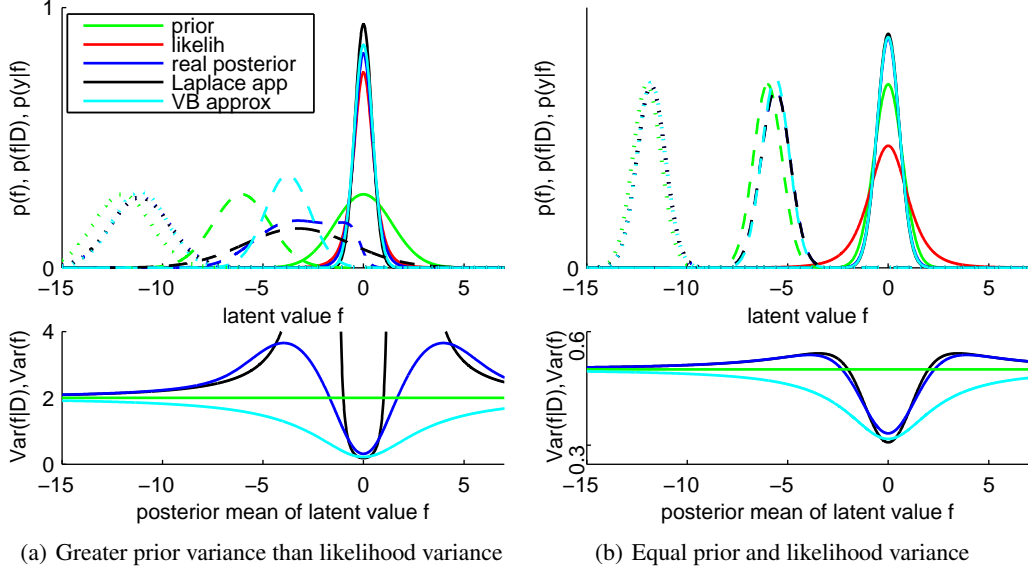

Figure 2: A comparison of the Laplace and VB approximation for $p(f|\theta, y)$ in the case of a single observation with the Student-$t$ likelihood and a Gaussian prior. The likelihood is centered at zero and the prior mean is altered. The upper plots show the probability density functions and the lower plots the variance of the true posterior and its approximations as a function of the posterior mean.

such as the VB in [7, 11]. An other concern is that the estimate of the posterior precision, $\mathbf{\Sigma}^{-1} = -\nabla\nabla \log p(\mathbf{f} \,|\, \mathbf{y}, \theta)|_{\mathbf{f}=\hat{\mathbf{f}}}$, is essentially uncontrolled. However, at a posterior mode $\hat{\mathbf{f}}$, the Hessian $\mathbf{\Sigma}^{-1}$ is always positive definite and in practice approximates the truth rather well according to our experiments. If the optimization for $\mathbf{f}$ ends up in a saddle point or the mode is very flat, $\mathbf{\Sigma}^{-1}$ may be close to singular, which leads to problems in the implementation. In this section, we will discuss these issues with simple examples and address the implementation in the section 4.

Consider a single observation $y_i = 0$ from a Student-$t$ distribution with a Gaussian prior for its mean, $f_i$. The behavior of the true posterior, the Laplace approximation, and VB as a function of prior mean are illustrated in the upper plots of the Figure 2. The dotted lines represent the situation, where the observation is a clear outlier in which case the posterior is very close to the prior (cf. section 2). The solid lines represent a situation where the prior and data agree, and the dashed lines represent a situation where the prior and data conflict moderately.

The posterior of the mean is unimodal if $\Sigma(f_i)^{-1} = \tau_i^{-2} + W(f_i) > 0$, for all $f_i \in \Re$, where $\tau_i^2$ is the prior variance and $W(f_i)$ is the Hessian of the negative log likelihood at $f_i$ (see equations (3) and (4)). With $\nu$ and $\sigma$ fixed, $W(f_i)$ reaches its (negative) minimum at $|y_i - f_i| = \pm\sqrt{3\nu}\sigma$, where $\Sigma^{-1} = \tau_i^{-2} - (\nu + 1)/(8\nu\sigma^2)$. Therefore, the posterior distribution is unimodal if $\tau_i^{-2} > (\nu + 1)/(8\nu\sigma^2)$, or in terms of variances if $\text{Var}[y_i|f_i, \nu, \sigma]/\tau_i^2 > (\nu + 1)/(8(\nu - 2))$ (for $\nu > 2$). It follows that the most problematic situation for the Laplace approximation is when the prior is much wider than the likelihood. Then in the case of a moderate conflict ($|y_i - \hat{f}_i|$ is close to $\sqrt{3\nu}\sigma$) the posterior may be multimodal (see the Figure 2(a)), meaning that it is unclear whether the observation is an outlier or not. In this case, $W(f_i)$ is negative and $\Sigma^{-1}$ may be close to zero, which reflects uncertainty on the location. In the implementation this may lead to numerical problems but in practice, the problem becomes concrete only seldom as described in the section 4.

The negative values of $W$ relate to a decrease in the posterior precision compared to the prior precision. As long as the total precision remains positive it approximates the behavior of the true posterior rather well. The Student-$t$ likelihood leads to a decrease in the variance from prior to posterior only if the prior mean and the observation are consistent with each other as shown in the Figure 2. This behavior is not captured with the factorized VB approximation [7], where $\mathbf{W}$ in $q(\mathbf{f} \,|\, \theta, \mathbf{y})$ is replaced with a strictly positive diagonal that always increases the precision as illustrated in the Figure 2.

## 4 On the implementation

### 4.1 Posterior mode of the latent variables

The mode of the latent variables, $\hat{\mathbf{f}}$, can be found with general optimization methods such as the scaled conjugate gradients. The most robust and efficient method, however, proved to be the expectation maximization (EM) algorithm that utilizes the scale mixture representation of the Student-$t$ distribution

$$y_i | f_i \sim N(f_i, V_i) \tag{10}$$

$$V_i \sim \text{Inv-}\chi^2(\nu, \sigma^2) \tag{11}$$

where each observation has its own noise variance $V_i$ that is Inv-$\chi^2$ distributed. Following Gelman et al. [13], p. 456 the E-step of the algorithm consists of evaluating the expectation

$$E\left[\frac{1}{V_i} \Big| y_i, f_i^{\text{old}}, \nu, \sigma\right] = \frac{\nu + 1}{\nu\sigma^2 + (y_i - f_i^{\text{old}})^2}, \tag{12}$$

after which the latent variables are updated in the M-step as

$$\hat{\mathbf{f}}^{\text{new}} = (\mathbf{K}_{\text{f,f}}^{-1} + \mathbf{V}^{-1})^{-1}\mathbf{V}^{-1}\mathbf{y}, \tag{13}$$

where $\mathbf{V}^{-1}$ is a diagonal matrix of the expectations in (12). In practice, we do not invert $\mathbf{K}_{\text{f,f}}$ and, thus, $\hat{\mathbf{f}}$ is updated using the Woodbury-Sherman-Morrison [e.g. 16] lemma

$$\begin{aligned}\hat{\mathbf{f}}^{\text{new}} &= (\mathbf{K}_{\text{f,f}} - \mathbf{K}_{\text{f,f}}\mathbf{V}^{-1/2}\mathbf{B}^{-1}\mathbf{V}^{-1/2}\mathbf{K}_{\text{f,f}})\mathbf{V}^{-1}\mathbf{y} \\ &= \mathbf{K}_{\text{f,f}}\mathbf{a}\end{aligned} \tag{14}$$

where matrix $\mathbf{B} = \mathbf{I} + \mathbf{V}^{-1/2}\mathbf{K}_{\text{f,f}}\mathbf{V}^{-1/2}$. This is numerically more stable than directly inverting the covariance matrix, and gives as an intermediate result the vector $\mathbf{a} = \mathbf{K}_{\text{f,f}}^{-1}\hat{\mathbf{f}}$ for later use.

### 4.2 Approximate marginal likelihood

Rasmussen and Williams [12] discuss a numerically stable formulation to evaluate the approximate marginal likelihood and its gradients with a classification model. Their approach relies on $\mathbf{W}$ being non-negative, for which reason it requires some modification for our setting. With the Student-$t$ likelihood, we found the most stable formulation for (6) is

$$\log q(\mathbf{y}\,|\theta) = \log p(\mathbf{y}|\hat{\mathbf{f}}) - \frac{1}{2}\hat{\mathbf{f}}^{\text{T}}\mathbf{a} - \sum_{i=1}^{n}\log\mathbf{R}_{ii} + \sum_{i=1}^{n}\log\mathbf{L}_{ii}, \tag{15}$$

where $\mathbf{R}$ and $\mathbf{L}$ are the Cholesky decomposition of $\mathbf{K}_{\text{f,f}}$ and $\mathbf{\Sigma} = (\mathbf{K}_{\text{f,f}}^{-1} + \mathbf{W})^{-1}$, and $\mathbf{a}$ is obtained from the EM algorithm. The only problematic term is the last one, which is numerically unstable if evaluated directly. We could evaluate first $\mathbf{\Sigma} = \mathbf{K}_{\text{f,f}} - \mathbf{K}_{\text{f,f}}(\mathbf{W}^{-1} + \mathbf{K}_{\text{f,f}})^{-1}\mathbf{K}_{\text{f,f}}$, but this is in many cases even worse than the direct evaluation, since $\mathbf{W}^{-1}$ might have arbitrary large negative values. For this reason, we evaluate $\mathbf{L}\mathbf{L}^{\text{T}} = \mathbf{\Sigma}$ using a rank one Cholesky updates in a specific order. After $\mathbf{L}$ is found it can also be used in the predictive variance (8) and in the gradients of (6) with only minor modification to equations given in [12]. We write first the posterior covariance as

$$\mathbf{\Sigma} = (\mathbf{K}_{\text{f,f}}^{-1} + \mathbf{W})^{-1} = (\mathbf{K}_{\text{f,f}}^{-1} + \mathbf{e}_1\mathbf{e}_1^{\text{T}}W_{11} + \mathbf{e}_2\mathbf{e}_2^{\text{T}}W_{22} + ...\mathbf{e}_n\mathbf{e}_n^{\text{T}}W_{nn})^{-1}, \tag{16}$$

where $\mathbf{e}_i$ is the $i$th unit vector. The terms $\mathbf{e}_i\mathbf{e}_i^{\text{T}}W_{ii}$ are added iteratively and the Cholesky decomposition of $\mathbf{\Sigma}$ is updated accordingly. At the beginning $\mathbf{L} = \text{chol}(\mathbf{K}_{\text{f,f}})$, and at iteration step $i+1$ we use the rank one Cholesky update to find

$$\mathbf{L}^{(i+1)} = \text{chol}\left(\mathbf{L}^{(i)}(\mathbf{L}^{(i)})^{\text{T}} - s_i s_i^{\text{T}}\delta_i\right), \tag{17}$$

where $s_i$ is the $i$th column of $\mathbf{\Sigma}^{(i)}$ and $\delta_i = W_{ii}(\mathbf{\Sigma}_{ii}^{(i)})^{-1}/((\mathbf{\Sigma}_{ii}^{(i)})^{-1} + W_{ii})$. If $W_{ii}$ is positive we conduct a Cholesky downdate, and if $W_{ii} < 0$ and $(\mathbf{\Sigma}_{ii}^{(i)})^{-1} + W_{ii} > 0$ we have a Cholesky update which increases the covariance. The increase may be arbitrary large if $(\mathbf{\Sigma}_{ii}^{(i)})^{-1} \approx -W_{ii}$, but in

practice it can be limited. Problems arise also if $W_{ii} < 0$ and $(\mathbf{\Sigma}_{ii}^{(i)})^{-1} + W_{ii} \leq 0$, since then the resulting Cholesky downdate is not positive definite. This should not happen if $\hat{\mathbf{f}}$ is at local maxima, but in practice it may be in a saddle point or this happens because of numerical instability or the iterative framework to update the Cholesky decomposition. The problem is prevented by adding the diagonals in a decreasing order, that is, first the "normal" observations and last the outliers.

A single Cholesky update is analogous to the discussion in section 3.4 in that the posterior covariance is updated using the result of the previous iteration as a prior. If we added the negative $\mathbf{W}$ values at the beginning, $\Sigma_{ii}$, (the prior variance) could be so large that either $(\mathbf{\Sigma}_{ii}^{(i)})^{-1} + W_{ii} \leq 0$ or $(\mathbf{\Sigma}_{ii}^{(i)})^{-1} \approx -W_{ii}$, in which case the posterior covariance $\Sigma_{ii}^{(i+1)}$ could become singular or arbitrary large and lead to problems in the later iterations (compare to the dashed black line in the Figure 2(a)). Adding first the largest $W$ we reduce $\mathbf{\Sigma}$ so that negative values of $W$ are less problematic (compare to the dashed black line in the Figure 2(b)), and the updates are numerically more stable.

During the Cholesky updates, we cross-check with the condition $(\mathbf{\Sigma}_{ii}^{(i)})^{-1} + W_{ii} \geq 0$ that everything is fine. If the condition is not fulfilled our code prints a warning and replaces $W_{ii}$ with $-1/(2\mathbf{\Sigma}_{ii}^{(i)})$. This ensures that the Cholesky update will remain positive definite and doubles the marginal variance instead. However, in practice we never encountered any warnings in our experiments if the hyperparameters were initialized sensibly so that the prior was tight compared to the likelihood.

## 5    Relation to other work

Neal [6] implemented the Student-$t$ model for the Gaussian process via Markov chain Monte Carlo utilizing the scale mixture representation. However, the most similar approaches to the Laplace approximation are the VB approximation [7, 11] and the one in INLA [15]. Here we will shortly summarize them.

The difference between INLA and GP framework is that INLA utilizes Gaussian Markov random fields (GMRF) in place of the Gaussian process. The Gaussian approximation for $p(\mathbf{f} \,|\, \mathbf{y}, \theta)$ in INLA is the same as the Laplace approximation here with the covariance function replaced by a precision matrix. Rue et al. [15] derive the approximation for the log marginal posterior, $\log p(\theta | \mathbf{y})$, from

$$p(\theta| \mathbf{y}) \approx q(\theta| \mathbf{y}) \propto \frac{p(\mathbf{y}, \mathbf{f}, \theta)}{q(\mathbf{f} \,|\theta, \mathbf{y})}\Big|_{\mathbf{f}=\hat{\mathbf{f}}} = \frac{p(\mathbf{y} \,|\, \mathbf{f})p(\mathbf{f} \,|\theta)p(\theta)}{q(\mathbf{f} \,|\theta, \mathbf{y})}\Big|_{\mathbf{f}=\hat{\mathbf{f}}}. \tag{18}$$

The proportionality sign is due to the fact that the normalization constant for $p(\mathbf{f}, \theta| \mathbf{y})$ is unknown. This is exactly the same as the approximation derived in the section 3.2. Taking the logarithm of (18) we end up in $\log q(\theta| \mathbf{y}) \propto \log q(\mathbf{y} \,|\theta) + \log p(\theta)$, where $\log q(\mathbf{y} \,|\theta)$ is given in (6).

In the variational approximation [7], the joint posterior of the latent variables and the scale parameters in the scale mixture representation (10)-(11) is approximated with a factorizing distribution $p(\mathbf{f}, \mathbf{V} | \mathbf{y}, \theta) \approx q(\mathbf{f})q(\mathbf{V})$, where $q(\mathbf{f}) = N(\mathbf{f} \,|\mathbf{m}, \mathbf{A})$ and $q(\mathbf{V}) = \Pi_{i=1}^n \text{Inv-}\chi^2(V_{ii}|\tilde{\nu}/2, \tilde{\sigma}^2/2)$, where $\tilde{\theta} = \{\mathbf{m}, \mathbf{A}, \tilde{\nu}, \tilde{\sigma}^2\}$ are the parameters of the variational approximation. The approximate distributions and the hyperparameters are updated in turns so that $\tilde{\theta}$ are updated with current estimate for $\theta$ and after that $\theta$ is updated with fixed $\tilde{\theta}$.

The variational approximation for the conditional posterior is $p(\mathbf{f} \,|\, \mathbf{y}, \hat{\theta}, \hat{\mathbf{V}}) \approx N(\mathbf{f} \,|\mathbf{m}, \mathbf{A})$. Here, $\mathbf{A} = (\mathbf{K}_{\mathrm{f,f}}^{-1} + \hat{\mathbf{V}}^{-1})^{-1}$, and the iterative search for the posterior parameters $\mathbf{m}$ and $\mathbf{A}$ is the same as the EM algorithm described in section 4 except that the update of $\text{E}\left[V_{ii}^{-1}\right]$ in (12) is replaced with $\text{E}\left[V_{ii}^{-1}\right] = (\nu+1)/(\sigma^2 + A_{ii}^{\text{old}} + (y_i - m_i^{\text{old}})^2)$. Thus, the Laplace and the variational approximation are very similar. In practice, the posterior mode, $\mathbf{m}$, is very close to the mode $\hat{\mathbf{f}}$, and the main difference between the approximations is in the covariance and the hyperparameter estimates.

In the variational approximation $\hat{\theta}$ is searched by maximizing the variational lower bound

$$\mathcal{V} = E_{q(\mathbf{f}, \mathbf{V} | \mathbf{y}, \theta)}\left[\log \frac{p(\mathbf{y}, \mathbf{f}, \mathbf{V}, \theta)}{q(\mathbf{f} \,|\, \mathbf{y}, \theta)q(\mathbf{V} | \mathbf{y}, \theta)}\right] = E_{q(\mathbf{f}, \mathbf{V} | \mathbf{y}, \theta)}\left[\log \frac{p(\mathbf{y} \,|\, \mathbf{f}, \mathbf{V})p(\mathbf{f} \,|\theta)p(\mathbf{V} |\theta)p(\theta)}{q(\mathbf{f}, \mathbf{V} | \mathbf{y}, \theta)}\right], \tag{19}$$

where we have made visible the implicit dependence of the approximations $q(\mathbf{f})$ and $q(\mathbf{V})$ to the data and hyperparameters, and included prior for $\theta$. The variational lower bound is similar to the ap-

Table 1: The RMSE and NLP statistics on the experiments.

|  | The RMSE error | | | | The NLP statistics | | | |
|  | Neal | Friedman | Housing | Concrete | Neal | Friedman | Housing | Concrete |
|---|---|---|---|---|---|---|---|---|
| G | 0.393 | 0.324 | 0.324 | 0.230 | 0.254 | 0.227 | 1.249 | 0.0642 |
| T-lapl | 0.028 | 0.220 | 0.289 | 0.231 | -2.181 | -0.16 | 0.080 | -0.116 |
| T-vb | 0.029 | 0.220 | 0.294 | 0.212 | -2.228 | -0.049 | 0.091 | -0.132 |
| T-mcmc | 0.055 | 0.253 | 0.287 | 0.197 | -1.907 | -0.106 | 0.029 | -0.241 |

proximate log marginal posterior (18). Only the point estimate $\hat{\mathbf{f}}$ is replaced with averaging over the approximating distribution $q(\mathbf{f}, \mathbf{V} \,|\, \mathbf{y}, \theta)$. The other difference is that in the Laplace approximation the scale parameters $\mathbf{V}$ are marginalized out and it approximates directly $p(\mathbf{f} \,|\, \mathbf{y}, \theta)$.

# 6   Experiments

We studied four data sets: 1) Neal data [6] with 100 data points and one input shown in Figure 1. 2) Friedman data with a nonlinear function of 10 inputs, from which we generated 10 data sets with 100 training points including 10 randomly selected outliers as described by Kuss [7], p. 83. 3) The Boston housing data that summarize median house prices in Boston metropolitan area for 506 data points and 13 input variables [7]. 4) Concrete data that summarize the quality of concrete casting as a function of 27 variables for 215 measurements [17]. In earlier experiments, the Student-$t$ model has worked better than the Gaussian observation model in all of these data sets.

The predictive performance is measured with a root mean squared error (RMSE) and a negative log predictive density (NLP). With simulated data these are evaluated for a test set of 1000 latent variables. With real data we use 10-fold cross-validation. The compared observation models are Gaussian (G) and Student-$t$ (T). The Student-$t$ model is inferred using the Laplace approximation (lapl), VB (vb) [7] and full MCMC (mcmc) [6]. The Gaussian observation model, the Laplace approximation and VB are evaluated at $\hat{\theta}$, and in MCMC we sample $\theta$. INLA is excluded from the experiments since GMRF model can not be constructed naturally for these non-regularly distributed data sets. The results are summarized in the Table 1. The significance of the differences in performance is approximated using a Gaussian approximation for the distribution of the NLP and RMSE statistics [17]. The Student-$t$ model is significantly better than the Gaussian with higher than 95% probability in all other tests but in the RMSE with the concrete data. There is no significant difference between the Laplace approximation, VB and MCMC.

The inference time was the shortest with Gaussian observation model and the longest with the Student-$t$ model utilizing full MCMC. The Laplace approximation for the Student-$t$ likelihood took in average 50% more time than the Gaussian model, and VB was in average 8-10 times slower than the Laplace approximation. The reason for this is that in VB two sets of parameters, $\theta$ and $\tilde{\theta}$, are updated in turns, which slows down the convergence of hyperparameters. In the Laplace approximation we have to optimize only $\theta$. Figure 3 shows the mean and the variance of $p(\mathbf{f} \,|\, \hat{\theta}, \mathbf{y})$ for MCMC versus the Laplace approximation and VB. The mean of the Laplace approximation and VB match equally well the mean of the MCMC solution, but VB underestimates the variance more than the Laplace approximation (see also the figure 2). In the housing data, both approximations underestimate the variance remarkably for few data points (40 of 506) that were located as clusters at places where inputs, $\mathbf{x}$ are truncated along one or more dimension. At these locations, the marginal posteriors were slightly skew and their tails were rather heavy, and thus a Gaussian approximation presumably underestimates the variance.

The degrees of freedom of the Student-$t$ likelihood were optimized only in Neal data and Boston housing data using the Laplace approximation. In other data sets, there was not enough information to infer $\nu$ and it was set to 4. Optimizing $\nu$ was more problematic for VB than for the Laplace approximation probably because the factorized approximation makes it harder to identify $\nu$. The MAP estimates $\hat{\theta}$ found by the Laplace approximation and VB were slightly different. This is reasonable since the optimized functions (18) and (19) are also different.

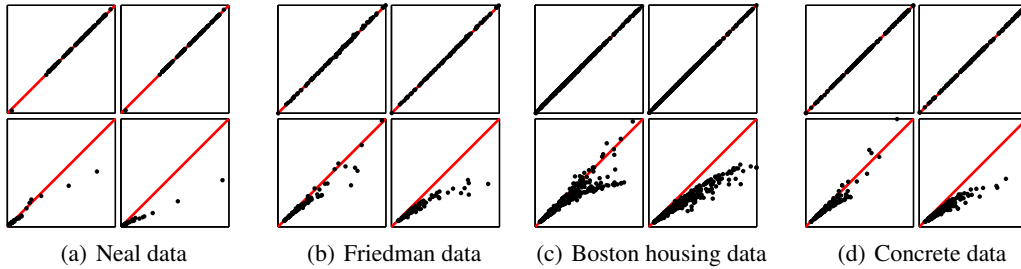

(a) Neal data      (b) Friedman data      (c) Boston housing data      (d) Concrete data

Figure 3: Scatter plot of the posterior mean and variance of the latent variables. Upper row consists means, and lower row variances. In each figure, left plot is for MCMC (x-axis) vs the Laplace approximation (y-axis) and the right plot is MCMC (x-axis) vs. VB (y-axis).

# 7 Discussion

In our experiments we found that the predictive performance of both the Laplace approximation and the factorial VB is similar with the full MCMC. Compared to the MCMC the Laplace approximation and VB estimate the posterior mean $E[\mathbf{f}\,|\hat{\theta}, \mathbf{y}]$ similarly but VB underestimates the posterior variance $\text{Var}[\mathbf{f}\,|\hat{\theta}, \mathbf{y}]$ more than the Laplace approximation. Optimizing the hyperparameters is clearly faster with the Laplace approximation than with VB.

Both the Laplace and the VB approximation estimate the posterior precision as a sum of a prior precision and a diagonal matrix. In VB the diagonal is strictly positive, whereas in the Laplace approximation the diagonal elements corresponding to outlying observations are negative. The Laplace approximation is closer to the reality in that respect since the outlying observations have a negative effect on the (true) posterior precision. This happens because VB minimizes $KL(q(\mathbf{f})q(\mathbf{V})||p(\mathbf{f}, \mathbf{V}))$, which requires that the $q(\mathbf{f}, \mathbf{V})$ must be close to zero whenever $p(\mathbf{f}, \mathbf{V})$ is (see for example [18]). Since a posteriori $\mathbf{f}$ and $\mathbf{V}$ are correlated, the marginal $q(\mathbf{f})$ underestimates the effect of marginalizing over the scale parameters. The Laplace approximation, on the other hand, tries to estimate directly the posterior $p(\mathbf{f})$ of the latent variables. Recently, Opper and Archambeau [19] discussed the relation between the Laplace approximation and VB, and proposed a variational approximation directly for the latent variables and tried it with a Cauchy likelihood (they did not perform extensive experiments though). Presumably their implementation would give better estimate for $p(\mathbf{f})$ than the factorized approximation. However, experiments on that respect are left for future.

The advantage of VB is that the objective function (19) is a rigorous lower bound for $p(\mathbf{y}\,|\theta)$, whereas the Laplace approximation (18) is not. However, the marginal posteriors $p(\mathbf{f}\,|\,\mathbf{y}, \theta)$ in our experiments (inferred with MCMC) were so close to Gaussian that the Laplace approximation $q(\mathbf{f}\,|\theta, \mathbf{y})$ should be very accurate and, thus, the approximation for $p(\theta|\,\mathbf{y})$ (18) should also be close to the truth (see also justifications in [15]).

In recent years the expectation propagation (EP) algorithm [20] has been demonstrated to be very accurate and efficient method for approximate inference in many models with factorizing likelihoods. However, the Student-$t$ likelihood is problematic for EP since it is not log-concave, for which reason EPs estimate for the posterior covariance may become singular during the site updates [21]. The reason for this is that the variance parameters of the site approximations may become negative. As demonstrated with Laplace approximation here, this reflects the behavior of the true posterior. We assume that the problem can be overcome, but we are not aware of any work that would have solved this problem.

### Acknowledgments

This research was funded by the Academy of Finland, and the Graduate School in Electronics and Telecommunications and Automation (GETA). The first and second author thank also the Finnish Foundation for Economic and Technology Sciences - KAUTE, Finnish Cultural Foundation, Emil Aaltonen Foundation, and Finnish Foundation for Technology Promotion for supporting their post graduate studies.

# References

[1] Bruno De Finetti. The Bayesian approach to the rejection of outliers. In *Proceedings of the fourth Berkeley Symposium on Mathematical Statistics and Probability*, pages 199–210. University of California Press, 1961.

[2] A. Philip Dawid. Posterior expectations for large observations. *Biometrika*, 60(3):664–667, December 1973.

[3] Anthony O'Hagan. On outlier rejection phenomena in Bayes inference. *Royal Statistical Society. Series B.*, 41(3):358–367, 1979.

[4] Mike West. Outlier models and prior distributions in Bayesian linear regression. *Journal of Royal Statistical Society. Serires B.*, 46(3):431–439, 1984.

[5] John Geweke. Bayesian treatment of the independent Student-$t$ linear model. *Journal of Applied Econometrics*, 8:519–540, 1993.

[6] Radford M. Neal. Monte Carlo Implementation of Gaussian Process Models for Bayesian Regression and Classification. Technical Report 9702, Dept. of statistics and Dept. of Computer Science, University of Toronto, January 1997.

[7] Malte Kuss. *Gaussian Process Models for Robust Regression, Classification, and Reinforcement Learning*. PhD thesis, Technische Universität Darmstadt, 2006.

[8] Paul W. Goldberg, Christopher K.I. Williams, and Christopher M. Bishop. Regression with input-dependent noise: A Gaussian process treatment. In M. I. Jordan, M. J. Kearns, and S. A Solla, editors, *Advances in Neural Information Processing Systems 10*. MIT Press, Cambridge, MA, 1998.

[9] Andrew Naish-Guzman and Sean Holden. Robust regression with twinned gaussian processes. In J.C. Platt, D. Koller, Y. Singer, and S. Roweis, editors, *Advances in Neural Information Processing Systems 20*, pages 1065–1072. MIT Press, Cambridge, MA, 2008.

[10] Oliver Stegle, Sebastian V. Fallert, David J. C. MacKay, and Søren Brage. Gaussian process robust regression for noisy heart rate data. *Biomedical Engineering, IEEE Transactions on*, 55 (9):2143–2151, September 2008. ISSN 0018-9294. doi: 10.1109/TBME.2008.923118.

[11] Michael E. Tipping and Neil D. Lawrence. Variational inference for Student-$t$ models: Robust bayesian interpolation and generalised component analysis. *Neurocomputing*, 69:123–141, 2005.

[12] Carl Edward Rasmussen and Christopher K. I. Williams. *Gaussian Processes for Machine Learning*. The MIT Press, 2006.

[13] Andrew Gelman, John B. Carlin, Hal S. Stern, and Donald B. Rubin. *Bayesian Data Analysis*. Chapman & Hall/CRC, second edition, 2004.

[14] Christopher K. I. Williams and David Barber. Bayesian classification with Gaussian processes. *IEEE Transactions on Pattern Analysis and Machine Intelligence*, 20(12):1342–1351, 1998.

[15] Håvard Rue, Sara Martino, and Nicolas Chopin. Approximate Bayesian inference for latent Gaussian models by using integrated nested Laplace approximations. *Journal of Royal statistical Society B*, 71(2):1–35, 2009.

[16] David A. Harville. *Matrix Algebra From a Statistician's Perspective*. Springer-Verlag, 1997.

[17] Aki Vehtari and Jouko Lampinen. Bayesian model assessment and comparison using cross-validation predictive densities. *Neural Computation*, 14(10):2439–2468, 2002.

[18] Christopher M. Bishop. *Pattern Recognition and Machine Learning*. Springer Science +Business Media, LLC, 2006.

[19] Manfred Opper and Cédric Archambeau. The variational Gaussian approximation revisited. *Neural Computation*, 21(3):786–792, March 2009.

[20] Thomas Minka. *A family of algorithms for approximate Bayesian inference*. PhD thesis, Massachusetts Institute of Technology, 2001.

[21] Matthias Seeger. Bayesian inference and optimal design for the sparse linear model. *Journal of Machine Learning Research*, 9:759–813, 2008.

